# Bayesian video shot segmentation

**Nuno Vasconcelos**     **Andrew Lippman**
MIT Media Laboratory, 20 Ames St, E15-354, Cambridge, MA 02139,
{nuno,lip}@media.mit.edu,     http://www.media.mit.edu/~nuno

## Abstract

*Prior knowledge about video structure can be used both as a means to improve the performance of content analysis and to extract features that allow semantic classification. We introduce statistical models for two important components of this structure, shot duration and activity, and demonstrate the usefulness of these models by introducing a Bayesian formulation for the shot segmentation problem. The new formulations is shown to extend standard thresholding methods in an adaptive and intuitive way, leading to improved segmentation accuracy.*

## 1 Introduction

Given the recent advances on video coding and streaming technology and the pervasiveness of video as a form of communication, there is currently a strong interest in the development of techniques for browsing, categorizing, retrieving and automatically summarizing video. In this context, two tasks are of particular relevance: the decomposition of a video stream into its component units, and the extraction of features for the automatic characterization of these units. Unfortunately, current video characterization techniques rely on image representations based on low-level visual primitives (such as color, texture, and motion) that, while practical and computationally efficient, fail to capture most of the structure that is relevant for the perceptual decoding of the video. In result, it is difficult to design systems that are truly useful for naive users. Significant progress can only be attained by a deeper understanding of the relationship between the message conveyed by the video and the patterns of visual structure that it exhibits.

There are various domains where these relationships have been thoroughly studied, albeit not always from a computational standpoint. For example, it is well known by film theorists that the message strongly constrains the stylistic elements of the video [1, 6], which are usually grouped into two major categories: the *elements of montage* and the *elements of mise-en-scene*. Montage refers to the temporal structure, namely the aspects of film editing, while, mise-en-scene deals with spatial structure, i.e. the composition of each image, and includes variables such as the type of set in which the scene develops, the placement of the actors, aspects of lighting, focus, camera angles, and so on. Building computational models for these stylistic elements can prove useful in two ways: on one hand it will allow the extraction of *semantic features* enabling video characterization and classification much closer to that which people use than current descriptors based on texture properties or optical flow. On the other hand, it will provide constraints for the low-level analysis algorithms required to perform tasks such as video segmentation, keyframing, and so on.

The first point is illustrated by Figure 1 where we show how a collection of promotional trailers for commercially released feature films populates a 2-D feature space based on the most elementary characterization of montage and mise-en-scene: average shot duration vs. average shot activity[1]. Despite the coarseness of this characterization, it captures aspects that are important for semantic movie classification: close inspection of the genre assigned to each movie by the *motion picture association of America* reveals that in this space the movies cluster by genre!

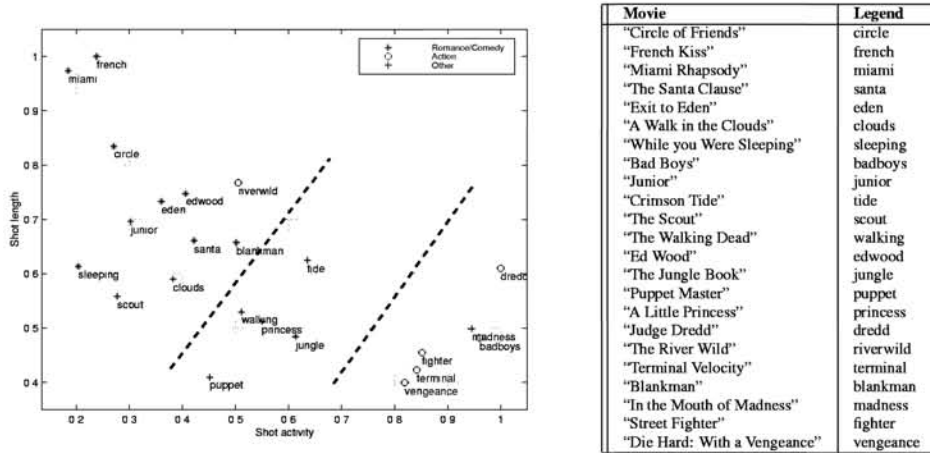

| Movie | Legend |
|---|---|
| "Circle of Friends" | circle |
| "French Kiss" | french |
| "Miami Rhapsody" | miami |
| "The Santa Clause" | santa |
| "Exit to Eden" | eden |
| "A Walk in the Clouds" | clouds |
| "While you Were Sleeping" | sleeping |
| "Bad Boys" | badboys |
| "Junior" | junior |
| "Crimson Tide" | tide |
| "The Scout" | scout |
| "The Walking Dead" | walking |
| "Ed Wood" | edwood |
| "The Jungle Book" | jungle |
| "Puppet Master" | puppet |
| "A Little Princess" | princess |
| "Judge Dredd" | dredd |
| "The River Wild" | riverwild |
| "Terminal Velocity" | terminal |
| "Blankman" | blankman |
| "In the Mouth of Madness" | madness |
| "Street Fighter" | fighter |
| "Die Hard: With a Vengeance" | vengeance |

Figure 1: Shot activity vs. duration features. The genre of each movie is identified by the symbol used to represent the movie in the plot.

In this paper, we concentrate on the second point, i.e. how the structure exhibited by Figure 1 can be exploited to improve the performance of low-level processing tasks such as shot segmentation. Because knowledge about the video structure is a form of prior knowledge, Bayesian procedures provide a natural way to accomplish this goal. We therefore introduce computational models for shot duration and activity and develop a Bayesian framework for segmentation that is shown to significantly outperform current approaches.

## 2  Modeling shot duration

Because shot boundaries can be seen as arrivals over discrete, non-overlapping temporal intervals, a Poisson process seems an appropriate model for shot duration [3]. However, events generated by Poisson processes have inter-arrival times characterized by the exponential density which is a monotonically decreasing function of time. This is clearly not the case for the shot duration, as can be seen from the histograms of Figure 2. In this work, we consider two alternative models, the Erlang and Weibull distributions.

### 2.1  The Erlang model

Letting $\tau$ be the time since the previous boundary, the Erlang distribution [3] is described by

$$\epsilon_{r,\lambda}(\tau) = \frac{\lambda^r \tau^{r-1} e^{-\lambda\tau}}{(r-1)!}. \tag{1}$$

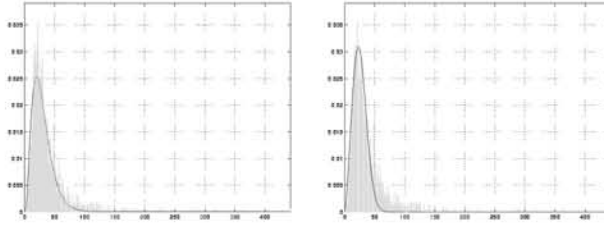

Figure 2: Shot duration histogram, and maximum likelihood fit obtained with the Erlang (left) and Weibull (right) distributions.

It is a generalization of the exponential density, characterized by two parameters: the order $r$, and the expected inter-arrival time $(1/\lambda)$ of the underlying Poisson process. When $r = 1$, the Erlang distribution becomes the exponential distribution. For larger values of $r$, it characterizes the time between the $r^{th}$ order inter-arrival time of the Poisson process. This leads to an intuitive explanation for the use of the Erlang distribution as a model of shot duration: for a given order $r$, the shot is modeled as a sequence of $r$ events which are themselves the outcomes of Poisson processes. Such events may reflect properties of the shot content, such as "setting the context" through a wide angle view followed by "zooming in on the details" when $r = 2$, or "emotional buildup" followed by "action" and "action outcome" when $r = 3$. Figure 2 presents a shot duration histogram, obtained from the training set to be described in section 5, and its maximum likelihood (ML) Erlang fit.

## 2.2 The Weibull model

While the Erlang model provides a good fit to the empirical density, it is of limited practical utility due to the constant arrival rate assumption [5] inherent to the underlying Poisson process. Because $\lambda$ is a constant, the expected rate of occurrence of a new shot boundary is the same if 10 seconds or 1 hour have elapsed since the occurrence of the previous one. An alternative models that does not suffer from this problem is the Weibull distribution [5], which generalizes the exponential distribution by considering an expected rate of arrival of new events that is a function of time $\tau$

$$\lambda(\tau) = \frac{\alpha \tau^{\alpha-1}}{\beta^{\alpha}},$$

and of the parameters $\alpha$ and $\beta$; leading to a probability density of the form

$$w_{\alpha,\beta}(\tau) = \frac{\alpha \tau^{\alpha-1}}{\beta^{\alpha}} \exp\left[ -\left(\frac{\tau}{\beta}\right)^{\alpha} \right]. \tag{2}$$

Figure 2 presents the ML Weibull fit to the shot duration histogram. Once again we obtain a good approximation to the empirical density estimate.

## 3  Modeling shot activity

The color histogram distance has been widely used as a measure of (dis)similarity between images for the purposes of object recognition [7], content-based retrieval [4], and temporal video segmentation [2]. A histogram is first computed for each image in the sequence and the distance between successive histograms is used as a measure of local activity. A standard metric for video segmentation [2] is the $L_1$ norm of the histogram difference,

$$\mathcal{D}(\mathbf{a}, \mathbf{b}) = \sum_{i=1}^{B} |a_i - b_i|, \tag{3}$$

where **a** and **b** are histograms of successive frames, and $B$ the number of histogram bins.

Statistical modeling of the histogram distance features requires the identification of the various states through which the video may progress. For simplicity, in this work we restrict ourselves to a video model composed of two states: "regular frames" ($\mathcal{S} = 0$) and "shot transitions" ($\mathcal{S} = 1$). The fundamental principles are however applicable to more complex models. As illustrated by Figure 3, for "regular frames" the distribution is asymmetric about the mean, always positive and concentrated near zero. This suggests that a mixture of Erlang distributions is an appropriate model for this state, a suggestion that is confirmed by the fit to the empirical density obtained with EM, also depicted in the figure. On the other hand, for "shot transitions" the fit obtained with a simple Gaussian model is sufficient to achieve a reasonable approximation to the empirical density. In both cases, a uniform mixture component is introduced to account for the tails of the distributions.

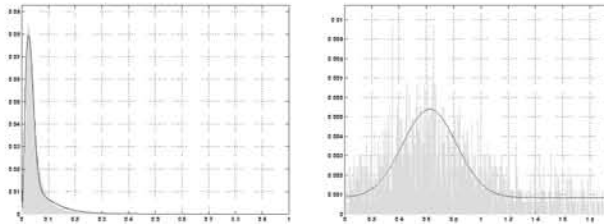

Figure 3: Left: Conditional activity histogram for regular frames, and best fit by a mixture with three Erlang and a uniform component. Right: Conditional activity histogram for shot transitions, and best fit by a mixture with a Gaussian and a uniform component.

## 4 A Bayesian framework for shot segmentation

Because shot segmentation is a pre-requisite for virtually any task involving the understanding, parsing, indexing, characterization, or categorization of video, the grouping of video frames into shots has been an active topic of research in the area of multimedia signal processing. Extensive evaluation of various approaches has shown that simple thresholding of histogram distances performs surprisingly well and is difficult to beat [2]. In this work, we consider an alternative formulation that regards the problem as one of statistical inference between two hypothesis:

- $\mathcal{H}_0$: no shot boundary occurs between the two frames under analysis ($\mathcal{S} = 0$),
- $\mathcal{H}_1$: a shot boundary occurs between the two frames ($\mathcal{S} = 1$),

for which the optimal decision is provided by a likelihood ratio test where $\mathcal{H}_1$ is chosen if

$$\mathcal{L} = \log \frac{P(\mathcal{D}|\mathcal{S} = 1)}{P(\mathcal{D}|\mathcal{S} = 0)} > 0, \tag{4}$$

and $\mathcal{H}_0$ is chosen otherwise. It is well known that standard thresholding is a particular case of this formulation, in which both conditional densities are assumed to be Gaussians with the same covariance. From the discussion in the previous section, it is clear that this does not hold for real video. One further limitation of the thresholding model is that it does not take into account the fact that the likelihood of a new shot transition is dependent on how much time has elapsed since the previous one. On the other hand, the statistical formulation can easily incorporate the shot duration models developed in section 2.

## 4.1 Notation

Because video is a discrete process, characterized by a given frame rate, shot boundaries are not instantaneous, but last for one frame period. To account for this, states are defined over time intervals, i.e. instead of $\mathcal{S}_t = 0$ or $\mathcal{S}_t = 1$, we have $\mathcal{S}_{t,t+\delta} = 0$ or $\mathcal{S}_{t,t+\delta} = 1$, where t is the start of a time interval, and $\delta$ its duration. We designate the features observed during the interval $[t, t + \delta]$ by $\mathcal{D}_{t,t+\delta}$.

To simplify the notation, we reserve $t$ for the temporal instant at which the last shot boundary has occurred and make all temporal indexes relative to this instant. I.e. instead of $\mathcal{S}_{t+\tau,t+\tau+\delta}$ we write $\mathcal{S}_{\tau,\tau+\delta}$, or simply $\mathcal{S}_\delta$ if $\tau = 0$. Furthermore, we reserve the symbol $\delta$ for the duration of the interval between successive frames (inverse of the frame rate), and use the same notation for a simple frame interval and a vector of frame intervals (the temporal indexes being themselves enough to avoid ambiguity). I.e., while $\mathcal{S}_{\tau,\tau+\delta} = 0$ indicates that no shot boundary is present in the interval $[t + \tau, t + \tau + \delta]$, $\mathcal{S}_{\tau+\delta} = \mathbf{0}$ indicates that no shot boundary has occurred in any of the frames between $t$ and $t + \tau + \delta$. Similarly, $\mathcal{D}_{\tau+\delta}$ represents the vector of observations in $[t, t + \tau + \delta]$.

## 4.2 Bayesian formulation

Given that there is a shot boundary at time $t$ and no boundaries occur in the interval $[t, t+\tau]$, the posterior probability that the next shot change happens during the interval $[t+\tau, t+\tau+\delta]$ is, using Bayes rule,

$$P(\mathcal{S}_{\tau,\tau+\delta} = 1 | \mathcal{S}_\tau = \mathbf{0}, \mathcal{D}_{\tau+\delta}) = \gamma P(\mathcal{D}_{\tau+\delta} | \mathcal{S}_\tau = \mathbf{0}, \mathcal{S}_{\tau,\tau+\delta} = 1) P(\mathcal{S}_{\tau,\tau+\delta} = 1 | \mathcal{S}_\tau = \mathbf{0}),$$

where $\gamma$ is a normalizing constant. Similarly, the probability of no change in $[t+\tau, t+\tau+\delta]$ is

$$P(\mathcal{S}_{\tau,\tau+\delta} = 0 | \mathcal{S}_\tau = \mathbf{0}, \mathcal{D}_{\tau+\delta}) = \gamma P(\mathcal{D}_{\tau+\delta} | \mathcal{S}_{\tau+\delta} = \mathbf{0}) P(\mathcal{S}_{\tau,\tau+\delta} = 0 | \mathcal{S}_\tau = \mathbf{0}),$$

and the *posterior odds ratio* between the two hypothesis is

$$
\frac{P(\mathcal{S}_{\tau,\tau+\delta} = 1 | \mathcal{S}_\tau = \mathbf{0}, \mathcal{D}_{\tau+\delta})}{P(\mathcal{S}_{\tau,\tau+\delta} = 0 | \mathcal{S}_\tau = \mathbf{0}, \mathcal{D}_{\tau+\delta})} = \frac{P(\mathcal{D}_{\tau,\tau+\delta} | \mathcal{S}_{\tau,\tau+\delta} = 1)}{P(\mathcal{D}_{\tau,\tau+\delta} | \mathcal{S}_{\tau,\tau+\delta} = 0)} \frac{P(\mathcal{S}_{\tau,\tau+\delta} = 1 | \mathcal{S}_\tau = \mathbf{0})}{P(\mathcal{S}_{\tau,\tau+\delta} = 0 | \mathcal{S}_\tau = \mathbf{0})}
$$

$$
= \frac{P(\mathcal{D}_{\tau,\tau+\delta} | \mathcal{S}_{\tau,\tau+\delta} = 1)}{P(\mathcal{D}_{\tau,\tau+\delta} | \mathcal{S}_{\tau,\tau+\delta} = 0)} \frac{P(\mathcal{S}_{\tau,\tau+\delta} = 1, \mathcal{S}_\tau = \mathbf{0})}{P(\mathcal{S}_{\tau+\delta} = \mathbf{0})} \tag{5}
$$

where we have assumed that, given $\mathcal{S}_{\tau,\tau+\delta}$, $\mathcal{D}_{\tau,\tau+\delta}$ is independent of all other $\mathcal{D}$ and $\mathcal{S}$. In this expression, while the first term on the right hand side is the ratio of the conditional likelihoods of activity given the state sequence, the second term is simply the ratio of probabilities that there may (or not) be a shot transition $\tau$ units of time after the previous one. Hence, the shot duration density becomes a *prior* for the segmentation process. This is intuitive since knowledge about the shot duration is a form of prior knowledge about the structure of the video that should be used to favor segmentations that are more plausible.

Assuming further that $\mathcal{D}$ is stationary, defining $\Delta_\tau = [t + \tau, t + \tau + \delta]$, considering the probability density function $p(\tau)$ for the time elapsed until the first scene change after $t$, and taking logarithms, leads to a *log posterior odds ratio* $\mathcal{L}_{post}$ of the form

$$
\mathcal{L}_{post} = \log \frac{P(\mathcal{D}_{\Delta_\tau} | \mathcal{S}_{\Delta_\tau} = 1)}{P(\mathcal{D}_{\Delta_\tau} | \mathcal{S}_{\Delta_\tau} = 0)} + \log \frac{\int_\tau^{\tau+\delta} p(\alpha) d\alpha}{\int_{\tau+\delta}^\infty p(\alpha) d\alpha}. \tag{6}
$$

The optimal answer to the question if a shot change occurs or not in $[t + \tau, t + \tau + \delta]$ is thus to declare that a boundary exists if

$$
\log \frac{P(\mathcal{D}_{\Delta_\tau} | \mathcal{S}_{\Delta_\tau} = 1)}{P(\mathcal{D}_{\Delta_\tau} | \mathcal{S}_{\Delta_\tau} = 0)} \geq \log \frac{\int_{\tau+\delta}^\infty p(\alpha) d\alpha}{\int_\tau^{\tau+\delta} p(\alpha) d\alpha} = \mathcal{T}(\tau), \tag{7}
$$

and that there is no boundary otherwise. Comparing this with (4), it is clear that the inclusion of the shot duration prior transforms the fixed thresholding approach into an adaptive one, where the threshold depends on how much time has elapsed since the previous shot boundary.

### 4.2.1 The Erlang model

It can be shown that, under the Erlang assumption,

$$\int_a^b \epsilon_{r,\lambda}(\tau)d\tau = \frac{1}{\lambda}\sum_{i=1}^r [\epsilon_{i,\lambda}(a) - \epsilon_{i,\lambda}(b)] \tag{8}$$

and the threshold of (7) becomes

$$\mathcal{T}_\epsilon(\tau) = \log \frac{\sum_{i=1}^r \epsilon_{i,\lambda}(\tau+\delta)}{\sum_{i=1}^r [\epsilon_{i,\lambda}(\tau) - \epsilon_{i,\lambda}(\tau+\delta)]}. \tag{9}$$

Its variation over time is presented in Figure 4. While in the initial segment of the shot, the threshold is large and shot changes are unlikely to be accepted, the threshold decreases as the scene progresses increasing the likelihood that shot boundaries will be declared.

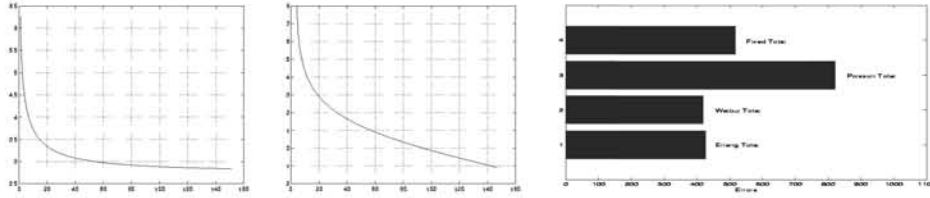

Figure 4: Temporal evolution of the Bayesian threshold for the Erlang (left) and Weibull (center) priors. Right: Total number of errors for all thresholds.

Even though, qualitatively, this is behavior that what one would desire, a closer observation of the figure reveals the major limitation of the Erlang prior: its steady-state behavior. Ideally, in addition to decreasing monotonically over time, the threshold should not be lower bounded by a positive value as this may lead to situations in which its steady-state value is high enough to miss several consecutive shot boundaries. This limitation is a consequence of the constant arrival rate assumption discussed in section 2 and can be avoided by relying instead on the Weibull prior.

### 4.2.2 The Weibull model

It can be shown that, under the Weibull assumption,

$$\int_a^b w_{\alpha,\beta}(\tau)d\tau = \exp\left[-\left(\frac{a}{\beta}\right)^\alpha\right] - \exp\left[-\left(\frac{b}{\beta}\right)^\alpha\right], \tag{10}$$

from which

$$\mathcal{T}_w(\tau) = -\log\left\{\exp\left[\frac{(\tau+\delta)^\alpha - \tau^\alpha}{\beta^\alpha}\right] - 1\right\}. \tag{11}$$

As illustrated by Figure 4, unlike the threshold associated with the Erlang prior, $\mathcal{T}_w(\tau)$ tends to $-\infty$ when $\tau$ grows without bound. This guarantees that a new shot boundary will always be found if one waits long enough. In summary, both the Erlang and the Weibull prior lead to adaptive thresholds that are more intuitive than the fixed threshold commonly employed for shot segmentation.

# 5 Segmentation Results

The performance of Bayesian shot segmentation was evaluated on a database containing the promotional trailers of Figure 1. Each trailer consists of 2 to 5 minutes of video and the total number of shots in the database is 1959. In all experiments, performance was evaluated by the *leave-one-out* method. Ground truth was obtained by manual segmentation of all the trailers.

We evaluated the performance of Bayesian models with Erlang, Weibull and Poisson shot duration priors and compared them against the best possible performance achievable with a fixed threshold. For the latter, the optimal threshold was obtained by brute-force, i.e. testing several values and selecting the one that performed best. Error rates for all priors are shown in Figure 4 where it is visible that, while the Poisson prior leads to worse accuracy than the static threshold, both the Erlang and the Weibull priors lead to significant improvements. The Weibull prior achieves the overall best performance decreasing the error rate of the static threshold by 20%.

The reasons for the improved performance of Bayesian segmentation are illustrated by Figure 5, which presents the evolution of the thresholding process for a segment from one of the trailers in the database ("blankman"). Two thresholding approaches are depicted: Bayesian with the Weibull prior, and standard fixed thresholding. The adaptive behavior of the Bayesian threshold significantly increases the robustness against spurious peaks of the activity metric originated by events such as very fast motion, explosions, camera flashes, etc.

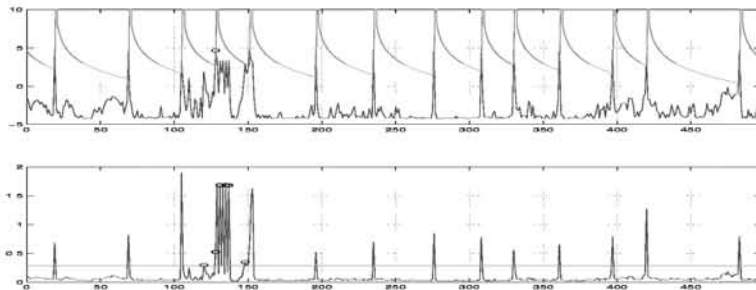

Figure 5: An example of the thresholding process. Top: Bayesian. The likelihood ratio and the Weibull threshold are shown. Bottom: Fixed. Histogram distances and optimal threshold (determined by leave-one-out using the remainder of the database) are presented. Errors are indicated by circles.

## Footnotes

[1]The activity features are described in section 3.

# References

[1] D. Bordwell and K. Thompson. *Film Art: an Introduction*. McGraw-Hill, 1986.

[2] J. Boreczky and L. Rowe. Comparison of Video Shot Boundary Detection Techniques. In *Proc. SPIE Conf. on Visual Communication and Image Processing*, 1996.

[3] A. Drake. *Fundamentals of Applied Probability Theory*. McGraw-Hill, 1987.

[4] W. Niblack et al. The QBIC project: Querying images by content using color, texture, and shape. In *Storage and Retrieval for Image and Video Databases*, pages 173–181, SPIE, Feb. 1993, San Jose, California.

[5] R. Hogg and E. Tanis. *Probability and Statistical Inference*. Macmillan, 1993.

[6] K. Reisz and G. Millar. *The Technique of Film Editing*. Focal Press, 1968.

[7] M. Swain and D. Ballard. Color Indexing. *International Journal of Computer Vision*, Vol. 7(1):11–32, 1991.
